# Bayesian time series classification

**Peter Sykacek**
Department of Engineering Science
University of Oxford
Oxford, OX1 3PJ, UK
*psyk@robots.ox.ac.uk*

**Stephen Roberts**
Department of Engineering Science
University of Oxford
Oxford, OX1 3PJ, UK
*sjrob@robots.ox.ac.uk*

## Abstract

This paper proposes an approach to classification of adjacent segments of a time series as being either of $K$ classes. We use a hierarchical model that consists of a feature extraction stage and a generative classifier which is built on top of these features. Such two stage approaches are often used in signal and image processing. The novel part of our work is that we link these stages probabilistically by using a *latent feature space*. To use *one* joint model is a Bayesian requirement, which has the advantage to fuse information according to its certainty.

The classifier is implemented as hidden Markov model with Gaussian and Multinomial observation distributions defined on a suitably chosen representation of autoregressive models. The Markov dependency is motivated by the assumption that successive classifications will be correlated. Inference is done with Markov chain Monte Carlo (MCMC) techniques. We apply the proposed approach to synthetic data and to classification of EEG that was recorded while the subjects performed different cognitive tasks. All experiments show that using a latent feature space results in a significant improvement in generalization accuracy. Hence we expect that this idea generalizes well to other hierarchical models.

## 1   Introduction

Many applications in signal or image processing are hierarchical in the sense that a probabilistic model is built on top of variables that are the coefficients of some feature extraction technique. In this paper we consider a particular problem of that kind, where a Gaussian and Multinomial observation hidden Markov model (GMOHMM) is used to discriminate coefficients of an Auto Regressive (AR) process as being either of $K$ classes. Bayesian inference is known to give reasonable results when applied to AR models ([RF95]). The situation with classification is similar, see for example the seminal work by [Nea96] and [Mac92]. Hence we may expect to get good results if we apply Bayesian techniques to both stages of the decision process separately. However this is suboptimal since it meant to establish a no probabilistic link between feature extraction and classification. Two arguments suggest the building of *one* probabilistic model which combines feature extraction and classification:

- Since there is a probabilistic link, the generative classifier acts as a prior for fea-

ture extraction. The advantage of using this prior is that it naturally encodes our knowledge about features as obtained from training data and other sensors. Obviously this is the only setup that is consistent with Bayesian theory ([BS94]).

- Since all inferences are obtained from marginal distributions, information is combined according to its certainty. Hence we expect to improve results since information from different sensors is fused in an optimal manner.

## 2 Methods

### 2.1 A Gaussian and Multinomial observation hidden Markov model

As we attempt to classify adjacent segments of a time series, it is very likely that we find correlations between successive class labels. Hence our model has a hidden Markov model ([RJ86]) like architecture, with diagonal Gaussian observation models for continuous variables and Multinomial observation models for discrete variables. We call the architecture a Gaussian and Multinomial observation hidden Markov model or GMOHMM for short. Contrary to the classical approach, where each class is represented by its own trained HMM, our model has class labels which are child nodes of the hidden state variables. Figure 1 shows the directed acyclic graph (DAG) of our model. We use here the convention found in [RG97], where circular nodes are latent and square nodes are observed variables.

#### 2.1.1 Quantities of interest

We regard all variables in the DAG that represent the probabilistic model of the time series as quantities of interest. These are the hidden states, $d_i$, the variables of the latent feature space, $\varphi_{i,s}$, $I_{i,s}$ and $\lambda_{i,s}$, the class labels, $t_i$, the sufficient statistics of the AR process, $s_{i,s}$, and the segments of the time series, $\mathcal{X}_{i,s}$. The DAG shows the observation model only for the $i$-th state. We have $s$ latent feature variables, $\varphi_{i,s}$, which represent the coefficients of the preprocessing model for of the $i$-th segment at sensor $s$. The state conditional distributions, $p(\varphi_{i,s}|d_i)$, are modeled by diagonal Gaussians. Variable $I_{i,s}$ is the latent model indicator which represents the model order of the preprocessing model and hence the dimension of $\varphi_{i,s}$. The corresponding observation model $p(I_{i,s}|d_i)$ is a Multinomial-one distribution. The third observation, $t_i$, represents the class label of the $i$-th segment. The observation model for $p(t_i|d_i)$ is again a Multinomial-one distribution. Note that depending on whether we know the class label or not, $t_i$ can be a latent variable or observed. The child node of $\varphi_{i,s}$ and $I_{i,s}$ is the observed variable $s_{i,s}$, which represents a sufficient statistics of the corresponding time series segment. The proposed approach requires to calculate the likelihoods $p(\mathcal{X}_{i,s}|\varphi_{i,s}, I_{i,s}) \ \forall \ i,s$ repeatedly. Hence using the sufficient statistics is a computational necessity. Finally we use $\lambda_{i,s}$ to represent the precision of the residual noise model. The noise level is a nuisance parameter which is integrated over.

#### 2.1.2 Model coefficients

Since we integrate over all unknown quantities, there is no conceptual difference between model coefficients and the variables described above. However there is a qualitative difference. Model parameters exist only once for the entire GMOHMM, whereas there is an individual quantity of interest for every segment $i$. Furthermore the model coefficients are only updated during model inference whereas all quantities of interest are updated during model inference and for prediction. We have three different prior counts, $\delta_T$, $\delta_W$ and $\delta_P$, which define the Dirichlet priors of the corresponding probabilities. Variable $\boldsymbol{T}$ denotes the transition probabilities, that is $P(d_{i+1}|d_i) = \boldsymbol{T}_{d_i}$. The model assumes a stationary hidden state sequence. This allows us to obtain the unconditional prior probability of states $d_i$ from the recurrence relation $\boldsymbol{P}_T(d_i) = \boldsymbol{T}\boldsymbol{P}(d_{i-1})$. The prior probability of the first hidden

state, $\boldsymbol{P}_T(d_1)$, is therefore the normalized eigenvector of the transition probability matrix $\boldsymbol{T}$ that corresponds to the eigenvalue 1. Variable $\boldsymbol{W}$ represents the probabilities of class $t_i$, $P(t_i|d_i) = \boldsymbol{W}_{d_i}$, which are conditional on $d_i$ as well. The prior probabilities for observing the model indicator $I_{i,s}$ are represented by $\boldsymbol{P}_s$. The probability $P(I_{i,s}|d_i) = \boldsymbol{P}_{s,d_i}$ is again conditional on the state $d_i$. As was mentioned above, $I_{i,s}$ represents the model order of the time series model. Hence another interpretation of $\boldsymbol{P}_s$ is that of state dependent prior probabilities for observing particular model orders. The observation models for $\boldsymbol{\varphi}_{i,s}$ are dynamic mixtures of Gaussians, with one model for each sensor $s$. Variables $\boldsymbol{\mu}_s$ and $\boldsymbol{\Sigma}_s$ represent the coefficients of all Gaussian kernels. Hence $p(\boldsymbol{\varphi}_{i,s}|\boldsymbol{\mu}_s, \boldsymbol{\Sigma}_s, d_i, I_{i,s})$ is a $I_{i,s}$-variate Gaussian distribution. Another interpretation is that the discrete indicator variables $d_i$ and $I_{i,s}$ determine together with $\boldsymbol{\mu}_s$ and $\boldsymbol{\Sigma}_s$ a Gaussian prior over $\boldsymbol{\varphi}_{i,s}$. The nodes $\boldsymbol{\kappa}_s$, $\boldsymbol{\xi}_s$, $\alpha_s$, $\boldsymbol{\beta}_s$, $g_s$ and $h_s$ define a hierarchical prior setting which is discussed below.

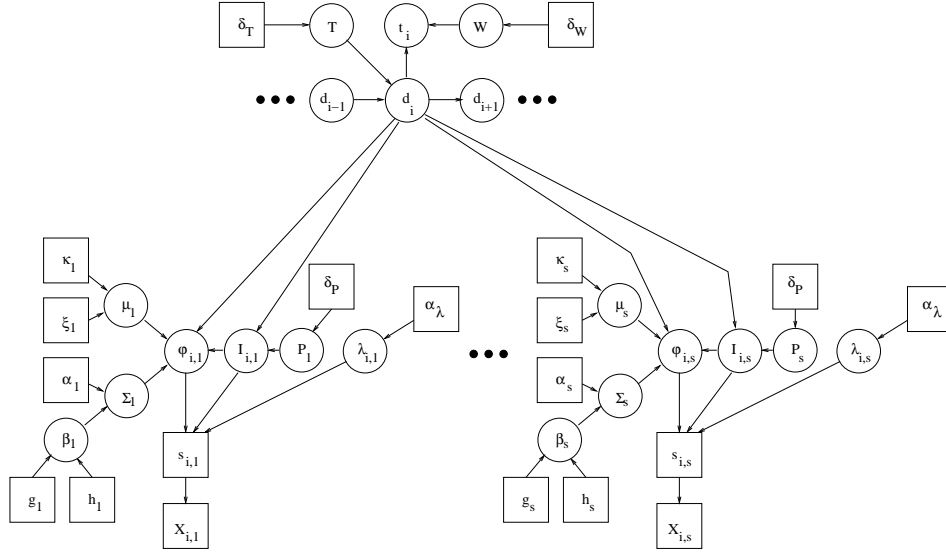

Figure 1: This figure illustrates the details of the proposed model as a directed acyclic graph. The graph shows the model parameters and all quantities of interest: $d_i$ denotes the hidden states of the HMM; $t_i$ are the class labels of the corresponding time series segments; $\boldsymbol{\varphi}_{i,s}$ are the latent coefficients of the time series model and $I_{i,s}$ the corresponding model indicator variables; $\lambda_{i,s}$ is the precision of the residual noise. For tractable inference, we extract from the time series $\mathcal{X}_{i,s}$ the sufficient statistics $s_{i,s}$. All other variables denote model coefficients: $\boldsymbol{T}$ are the transition probabilities; $\boldsymbol{W}$ are the probabilities for class $t_i$; $\boldsymbol{\mu}_s$ and $\boldsymbol{\Sigma}_s$ are mean vectors and covariance matrices of the Gaussian observation model for sensor $s$; and $P_s$ are the probabilities for observing $I_{i,s}$.

## 2.2  Likelihood and priors for the GMOHMM

Suppose that we are provided with $N$ segments of training data, $\mathcal{X} = \{\mathcal{X}_{i,s} \, \forall \, i, s\}$. The likelihood function of the GMOHMM parameters $\boldsymbol{w}$ is then obtained by summation over all possible sequences, $\Pi$, of latent states, $d_i$. The sums and integrals under the product make the likelihood function of Equation (1) highly nonlinear. This may be resolved by using Gibbs sampling [GG84], which uses tricks similar to those of the expectation maximization algorithm.

$$P(\mathcal{X}|\boldsymbol{w}) = \sum_{\Pi}\left(P(t_1,d_1)\prod_{s=1}^{S}\left(\sum_{I_{1,s}=0}^{I_{max}}\int\int p(\mathcal{X}_{1,s},\boldsymbol{\varphi}_{1,s},\lambda_{1,s},I_{1,s}|d_1)d\boldsymbol{\varphi}_{1,s}d\lambda_{1,s}\right)\right. \quad (1)$$

$$\left.\times\prod_{i=2}^{N}\left(P(t_i,d_i|d_{i-1})\prod_{s=1}^{S}\left(\sum_{I_{i,s}=0}^{I_{max}}\int\int p(\mathcal{X}_{i,s},\boldsymbol{\varphi}_{i,s},\lambda_{i,s},I_{i,s}|d_i)d\boldsymbol{\varphi}_{i,s}d\lambda_{i,s}\right)\right)\right).$$

Gibbs sampling requires that we obtain full conditional distributions[1] we can sample from. The conjugate priors are adopted from [RG97]. Below square brackets and index $j$ are used to denote a particular component of a vector or matrix. Each component mean, $\boldsymbol{\mu}_{s,d_i}$, is given a Gaussian prior: $\boldsymbol{\mu}_{s,d_i} \sim \mathcal{N}_1(\boldsymbol{\xi}_s,\boldsymbol{\kappa}_s^{-1})$, with $\boldsymbol{\xi}_s$ denoting the mean and $\boldsymbol{\kappa}_s^{-1}$ the inverse covariance matrix. As we use diagonal covariance matrices, we may give each diagonal element an independent Gamma prior: $\boldsymbol{\Sigma}_{s,d_i}[j,j]^{-1} \sim \Gamma(\alpha_s,\boldsymbol{\beta}_s[j])$, where $\alpha_s$ denotes the shape parameter and $\boldsymbol{\beta}_s[j]$ denotes the inverse scale parameter. The hyperparameter, $\beta_s$, gets a component wise Gamma hyper prior: $\boldsymbol{\beta}_s[j] \sim \Gamma(g_s,\boldsymbol{h}_s[j])$. The state conditional class probabilities, $\boldsymbol{W}_{d_i}$, get a Dirichlet prior: $\boldsymbol{W}_{d_i} \sim \mathcal{D}(\delta_W,..,\delta_W)$. The transition probabilities, $\boldsymbol{T}_{d_i}$, get a Dirichlet prior: $\boldsymbol{T}_{d_i} \sim \mathcal{D}(\delta_T,..,\delta_T)$. The probabilities for observing different model orders, $\boldsymbol{P}_{s,d_i}$, depend on the state $d_i$. Their prior is Dirichlet $\boldsymbol{P}_{s,d_i} \sim \mathcal{D}(\delta_P,..,\delta_P)$. The precision $\lambda_{i,s}$ gets a Jeffreys' prior, i.e. the scale parameter $a_\lambda$ is set to 0.

Values for $\alpha_s$ are between 1 and 2, $g_s$ is set between 0.2 and 1 and $\boldsymbol{h}_s[j]$ is typically between $1/R[j]^2$ and $10/R[j]^2$, with $R[j]$ denoting the input range of maximum likelihood estimates for $\boldsymbol{\varphi}_{i,s}[j] \ \forall\, i$. The mean, $\boldsymbol{\xi}_s$, is the midpoint of the maximum likelihood estimates $\boldsymbol{\varphi}_{i,s}[j] \ \forall\, i$. The inverse covariance matrix $\boldsymbol{\kappa}_s[j] = 1/R_s[j]^2$, where $R_s[j]$ is again the range of the estimates at sensor $s$. We set the prior counts $\delta_W$ and $\delta_T$ and $\delta_P$ to 1.

### 2.3 Sampling from the posterior

During model inference we need to update all unobserved variables of the DAG, whereas for predictions we update only the variables summarized in section 2.1.1. Most of the updates are done using the corresponding full conditional distributions, which have the same functional forms as the corresponding priors. These full conditionals follow closely from what was published previously in [Syk00], with some modifications necessary (see e.g. [Rob96]), because we need to consider the Markov dependency between successive hidden states. As the derivations of the full conditionals do not differ much from previous work, we will omit them here and instead concentrate on an illustration how to update the latent feature space, $(\boldsymbol{\varphi}_{i,s},I_{i,s}) \ \forall\, i,s$.

#### 2.3.1 A representation of the latent feature space

The AR model in Equation (2) is a linear regression model. We use $a^m_{I_{i,s}}$ to denote the AR coefficients, $I_{i,s}$ to denote the model order and $\epsilon[t]$ to denote a sample from the noise process, which we assume to be Gaussian with precision $\lambda_{i,s}$.

$$x[t] = -\sum_{m=1}^{I_{i,s}} a^m_{I_{i,s}} x[t-m] + \epsilon[t] \quad (2)$$

As is indicated by the subscript $I_{i,s}$, the value of the $m$-th AR coefficient depends on the model order. Hence AR coefficients are not a convenient representation of the latent feature

space. A much more convenient representation is provided by using reflection coefficients, $\rho$, (statistically speaking they are partial correlation coefficients), which relate to AR coefficients via the order recursive Levinson algorithm. Below we use vector notation and the symbol $\boldsymbol{a}_{I_{i,s}}^{\circlearrowleft}$ to denote the upside down version of the AR coefficient vector.

$$\boldsymbol{a}_{I_{i,s}+1} = \begin{bmatrix} \boldsymbol{a}_{I_{i,s}} + \rho_{(I_{i,s}+1)}\boldsymbol{a}_{I_{i,s}}^{\circlearrowleft} \\ \rho_{(I_{i,s}+1)} \end{bmatrix} \tag{3}$$

We expect to observe only such data that was generated from dynamically stable AR processes. For such processes, the latent density is defined on $\rho_{i,s} \in [-1,1]^{I_{i,s}} \subset \Re^{I_{i,s}}$. This is in contrast with the proposed DAG, where we use a finite Gaussian mixture as probabilistic model for the latent variable, which is is defined on $\boldsymbol{\varphi_{i,s}} \in \Re^{I_{i,s}}$. In order to avoid this mismatch, we reparameterise the space of reflection coefficients by applying $\mathrm{arctanh}$, to obtain a more convenient representation of the latent features.

$$\boldsymbol{\varphi}_{i,s} = \mathrm{arctanh}(\boldsymbol{\rho}_{i,s}) \tag{4}$$

### 2.3.2 Within dimensional updates

The within dimensional updates can be done with a conventional Metropolis Hastings step. Integrating out $\lambda_{i,s}$, we obtain a Student t distributed likelihood function of the AR coefficients. In order to obtain likelihood ratio 1, we propose from the multivariate Student-t distribution shown below, reparameterise in terms of reflection coefficients and apply the $\mathrm{arctanh}$ transformation.

$$\begin{aligned} \boldsymbol{\varphi}'_{i,s} &= \mathrm{arctanh}(\boldsymbol{\rho}(\boldsymbol{a}'_{i,s})) \\ \text{where} \\ \boldsymbol{a}'_{i,s} &\sim \mathcal{S}t_\nu(\hat{\boldsymbol{a}}, \boldsymbol{\Sigma}) \\ \text{with} \\ \hat{\boldsymbol{a}} &= \boldsymbol{A}^{-1}\boldsymbol{r} \\ \boldsymbol{\Sigma} &= \boldsymbol{A}^{-1}\frac{(R_0 - \boldsymbol{r}^T\boldsymbol{A}^{-1}\boldsymbol{r})}{2\nu} \\ \nu &= N - I_{i,s} \end{aligned} \tag{5}$$

The proposal uses $\boldsymbol{A}$ to denote the $I_{i,s}$-dimensional sample auto-covariance matrix, $R_0$ is the sample variance, $\boldsymbol{r} = [R_1, ..., R_{I_{i,s}+1}]^T$ is a vector of sample autocorrelations at lags 1 to $I_{i,s} + 1$ and N denotes the number of samples of the time series $\mathcal{X}_{i,s}$. The proposal in Equation (5) gives a likelihood ratio of 1. The corresponding acceptance probability is

$$a = \min\left(1, \frac{p(\boldsymbol{\varphi}'_{i,s})\left|\frac{\partial\boldsymbol{\varphi}'_{i,s}}{\partial\boldsymbol{a}'_{i,s}}\right|}{p(\boldsymbol{\varphi}_{i,s})\left|\frac{\partial\boldsymbol{\varphi}_{i,s}}{\partial\boldsymbol{a}_{i,s}}\right|}\right). \tag{6}$$

The determinant of the Jacobian arises because we transform the AR coefficients using Equations (3) and (4).

### 2.3.3 Updating model orders

Updating model orders requires us to sample across different dimensional parameter spaces. One way of doing this is by using the reversible jump MCMC which was recently proposed in [Gre95]. We implement the reversible jump move from parameter space $\mathcal{C}_{i,s,I_{i,s}}$ to parameter space $\mathcal{C}_{i,s,I_{i,s}+1}$ as partial proposal. That is we propose a reflection coefficient from a distribution that is conditional on the AR coefficient $\boldsymbol{a}_{i,s}$. Integrating

out the precision of the noise model $\lambda_{i,s}$ we obtain again a Student-t distributed likelihood. This suggests the following proposal:

$$\boldsymbol{\varphi}'_{i,s} = [\boldsymbol{\varphi}_{i,s}, \operatorname{arctanh}(\rho)] \tag{7}$$

where

$$\rho \sim \mathcal{S}t_\nu(\hat{\rho}, \sigma)$$

with

$$\hat{\rho} = -\frac{k_2}{k_1}$$

$$\sigma = \sqrt{\frac{1 - \hat{\rho}^2}{2(N-1)}}$$

$$\nu = N - 1$$

$$k_1 = R_0 + 2\boldsymbol{a}_{i,s}^T \boldsymbol{r}_0 + \boldsymbol{a}_{i,s}^T \boldsymbol{A}_0 \boldsymbol{a}_{i,s}$$

$$k_2 = R_{I+2} + 2\boldsymbol{r}_0^T \boldsymbol{a}_{i,s}^{\circlearrowleft} + \boldsymbol{a}_{i,s}^T \boldsymbol{A}_0 \boldsymbol{a}_{i,s}^{\circlearrowleft}.$$

Equation (7) makes use of the sufficient statistics of the $I_{i,s} + 1$-dimensional AR process, $s = \{N, R_0, .., R_{I+2}\}$. We use $N$ to denote the number of observations and $R_l$ to denote the estimated auto covariance at time lag $l$ to obtain $\boldsymbol{r}_0^T = [R_1, .., R_{I+1}]$ and $\boldsymbol{A}_0$ as $I_{i,s}$ dimensional sample covariance matrix. Assuming that the probability of proposing this move is independent of $I_{i,s}$, the proposal from $\mathcal{C}_{i,s,I_{i,s}}$ to $\mathcal{C}_{i,s,I_{i,s}+1}$ has acceptance probability

$$a = \min\left(1, \left(1 - \frac{k_1^2}{k_2^2}\right)^{-\frac{N-1}{2}} \sqrt{\frac{\pi}{2}} \frac{\Gamma(\frac{N-1}{2})}{\Gamma(\frac{N}{2})} \frac{p(\boldsymbol{\varphi}'_{i,s})p(I_{i,s}+1)}{p(\boldsymbol{\varphi}_{i,s})(1-\rho^2)p(I_{i,s})}\right). \tag{8}$$

If we attempt an update from $\mathcal{C}_{i,s,I_{i,s}+1}$ to $\mathcal{C}_{i,s,I_{i,s}}$, we have to invert the second argument of the $\min$ operation in Equation (8).

## 3 Experiments

Convergence of all experiments is analysed by applying the method suggested in [RL96] to the sequence of observed data likelihoods (equation (1), when filling in all variables).

### 3.1 Synthetic data

Our first evaluation uses synthetic data. We generate a first order Markov sequence as target labels (2 state values) with 200 samples used for training and 600 used for testing. Each sample is used as label of a segment with 200 samples from an auto regressive process. If the label is 1, we generate data using reflection coefficients $(0.9, -0.8, 0.5)$. If the label is 2, we use the model $(0.9, -0.7, 0.6)$. The driving noise has variance 1. Due to sampling effects we obtain a data set with Bayes error $> 0$. In order to make the problem more realistic, we use a second state sequence to replace 20% of the segments with white noise. These "artifacts" are not correlated with the class labels.

In order to assess the effect of using a latent feature space, we perform three different tests: In the first run we use conventional feature extraction with a third order model and estimates found with maximum likelihood; In a second run we use again a third order model but integrate over feature values; Finally the third test uses the proposed architecture with a prior over model order which is "flat" between 0 and 5.

When compared with conditioning on feature estimates, the latent features show increased likelihood. The likelihood gets even larger when we regard both the feature values and the

model orders of the preprocessing stage as random variables. As can be seen in figure 2, this effect is also evident when we look at the generalization probabilities which become larger as well. We explain this by sharper "priors" over feature values and model orders, which are due to the information provided by temporal context[2] of every segment. This reduces the variance of the observation models which in turn increases likelihoods and target probabilities. Table 1 shows that these higher probabilities correspond to a significant improvement in generalization accuracy.

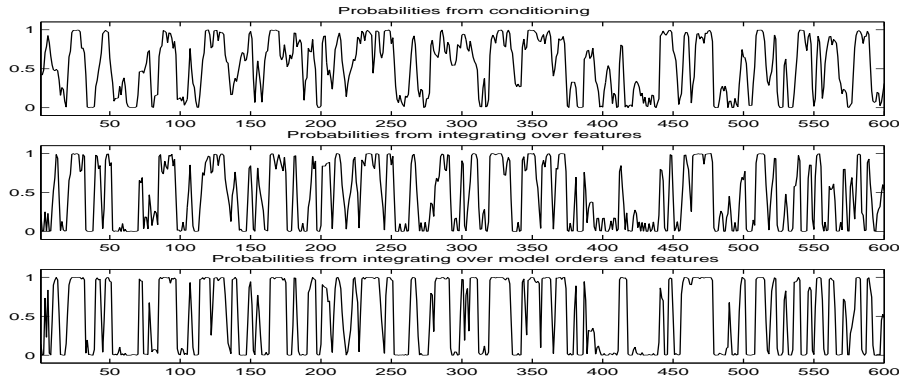

Figure 2: This figure shows the generalization probabilities obtained with different settings. We see that the class probabilities get larger when we regard features as random variables. This effect is even stronger when both the features and the model orders are random variables.

## 3.2 Classification of cognitive tasks

The data used in these experiments is EEG recorded from 5 young, healthy and *untrained* subjects while they perform different cognitive tasks. We classify 2 task pairings: auditory-navigation and left motor-right motor imagination. The recordings were taken from 3 electrode sites: T4, P4 (right tempero-parietal for spatial and auditory tasks), C3' , C3" (left motor area for right motor imagination) and C4' , C4" (right motor area for left motor imagination). The ground electrode was placed just lateral to the left mastoid process. The data were recorded using an ISO-DAM system (gain of $10^4$ and fourth order band pass filter with pass band between $0.1$ Hz and $100$ Hz). These signals were sampled with $384$ Hz and 12 bit resolution. Each cognitive experiment was performed 10 times for 7 seconds.

Classification uses again the same settings as with the synthetic problem. The summary in table 1 shows results obtained from 10 fold cross validation, where one experiment is used for testing whereas all remaining data is used for training. We observe again significantly improved results when we regard features and model orders as latent variables. The values in brackets are the significance levels for comparing integration of features with conditioning and full integration with integration over feature values only.

## 4 Discussion

We propose in this paper a novel approach to hierarchical time series processing which makes use of a latent feature representation. This understanding of features and model orders as random variables is a direct consequence of applying Bayesian theory. Empirical

Table 1: Generalization accuracies of different experiments

| experiment | conditioning | marginalize features | full integration |
|---|---|---|---|
| synthetic | 75.3% | 87.2% $(4.7 * 10^{-10})$ | 91.5% (0.002) |
| left vs. right motor | 75.9% | 79.5% $(6.2 * 10^{-4})$ | 81.43% (0.045) |
| auditory vs. navigation | 76.2% | 78.4%(0.02) | 84.5% $(2.3 * 10^{-9})$ |

evaluations show that theoretical arguments are confirmed by significant improvements in generalization accuracy. The only disadvantage of having a latent feature space is that all computations get more involved, since there are additional variables that have to be integrated over. However this additional complexity does *not* render the method intractable since the algorithm remains polynomial in the number of segments to be classified. Finally we want to point out that the improvements observed in our results can only be attributed to the idea of using a latent feature space. This idea is certainly not limited to time series classification and should generalize well to other hierarchical architectures.

## Acknowledgments

We want to express gratitude to Dr. Rezek, who made several valuable suggestions in the early stages of this work. We also want to thank Prof. Stokes, who provided us with the EEG recordings that were used in the experiments section. Finally we are also grateful for the valuable comments provided by the reviewers of this paper. Peter Sykacek is currently funded by grant Nr. F46/399 kindly provided by the BUPA foundation.

## Footnotes

[1]These are the distributions obtained when we condition on all other variables of the DAG.

[2]In a multi sensor setting there is spatial context as well.

## References

[BS94]  J. M. Bernardo and A. F. M. Smith. *Bayesian Theory*. Wiley, Chichester, 1994.

[GG84]  S. Geman and D. Geman. Stochastic relaxation, Gibbs distributions and the Bayesian restoration of images. *IEEE Transactions on Pattern Analysis and Machine Intelligence*, 6:721–741, 1984.

[Gre95]  P. J. Green. Reversible jump Markov chain Monte Carlo computation and Bayesian model determination. *Biometrika*, 82:711–732, 1995.

[Mac92]  D. J. C. MacKay. The evidence framework applied to classification networks. *Neural Computation*, 4:720–736, 1992.

[Nea96]  R. M. Neal. *Bayesian Learning for Neural Networks*. Springer, New York, 1996.

[RF95]  J. J. K. Ó Ruanaidh and W. J. Fitzgerald. *Numerical Bayesian Methods Applied to Signal Processing*. Springer-Verlag, New York, 1995.

[RG97]  S. Richardson and P. J. Green. On Bayesian analysis of mixtures with an unknown number of components. *Journal Royal Stat. Soc. B*, 59:731–792, 1997.

[RJ86]  L. R. Rabiner and B. H. Juang. An introduction to Hidden Markov Models. *IEEE ASSP Magazine*, 3(1):4–16, 1986.

[RL96]  A. E. Raftery and S. M. Lewis. Implementing MCMC. In W.R. Gilks, S. Richardson, and D.J. Spiegelhalter, editors, *Markov Chain Monte Carlo in practice*, chapter 7, pages 115–130. Chapman & Hall, London, Weinheim, New York, 1996.

[Rob96]  C. P. Robert. Mixtures of distributions: inference and estimation. In W. R. Gilks, S. Richardson, and D.J. Spiegelhalter, editors, *Markov Chain Mont Carlo in Practice*, pages 441–464. Chapman & Hall, London, 1996.

[Syk00]  P. Sykacek. On input selection with reversible jump Markov chain Monte Carlo sampling. In S.A. Solla, T.K. Leen, and K.-R. Müller, editors, *Advances in Neural Information Processing Systems 12*, pages 638–644, Boston, MA, 2000. MIT Press.
